# Why neuronal dynamics should control synaptic learning rules

**Jesper Tegnér**
Stockholm Bioinformatics Center
Dept. of Numerical Analysis
& Computing Science
Royal Institute for Technology
S-10044 Stockholm,Sweden
*jespert@nada.kth.se*

**Ádám Kepecs**
Volen Center for Complex Systems
Brandeis University
Waltham, MA 02454
*kepecs@brandeis.edu*

## Abstract

Hebbian learning rules are generally formulated as static rules. Under changing condition (e.g. neuromodulation, input statistics) most rules are sensitive to parameters. In particular, recent work has focused on two different formulations of spike-timing-dependent plasticity rules. Additive STDP [1] is remarkably versatile but also very fragile, whereas multiplicative STDP [2, 3] is more robust but lacks attractive features such as synaptic competition and rate stabilization. Here we address the problem of robustness in the additive STDP rule. We derive an adaptive control scheme, where the learning function is under fast dynamic control by postsynaptic activity to stabilize learning under a variety of conditions. Such a control scheme can be implemented using known biophysical mechanisms of synapses. We show that this adaptive rule makes the additive STDP more robust. Finally, we give an example how meta plasticity of the adaptive rule can be used to guide STDP into different type of learning regimes.

## 1 Introduction

Hebbian learning rules are widely used to model synaptic modification shaping the functional connectivity of neural networks [4, 5]. To ensure competition between synapses and stability of learning, constraints have to be added to correlational Hebbian learning rules [6]. Recent experiments revealed a mode of synaptic plasticity that provides new possibilities and constraints for synaptic learning rules [7, 8, 9]. It has been found that synapses are strengthened if a presynaptic spike precedes a postsynaptic spike within a short ($\approx$ 20 ms) time window, while the reverse spike order leads to synaptic weakening. This rule has been termed spike-timing dependent plasticity (STDP) [1]. Computational models highlighted how STDP combines synaptic strengthening and weakening so that learning gives rise to synaptic competition in a way that neuronal firing rates are stabilized.

Recent modeling studies have, however, demonstrated that whether an STDP type

rule results in competition or rate stabilization depends on exact formulation of the weight update scheme [3, 2]. Sompolinsky and colleagues [2] introduced a distinction between additive and multiplicative weight updating in STDP. In the additive version of an STDP update rule studied by Abbott and coworkers [1, 10], the magnitude of synaptic change is independent on synaptic strength. Here, it is necessary to add hard weight bounds to stabilize learning. For this version of the rule (aSTDP), the steady-state synaptic weight distribution is bimodal. In sharp contrast to this, using a multiplicative STDP rule where the amount of weight increase scales inversely with present weight size produces neither synaptic competition nor rate normalization [3, 2]. In this multiplicative scenario the synaptic weight distribution is unimodal. Activity-dependent synaptic scaling has recently been proposed as a separate mechanism to ensure synaptic competition operating on a slow (days) time scale [3]. Experimental data as of today is not yet sufficient to determine the circumstances under which the STDP rule is additive or multiplicative.

In this study we examine the stabilization properties of the additive STDP rule. In the first section we show that the aSTDP rule normalizes postsynaptic firing rates only in a limited parameter range. The critical parameter of aSTDP becomes the ratio ($\alpha$) between the amount of synaptic depression and potentiation. We show that different input statistics necessitate different $\alpha$ ratios for aSTDP to remain stable. This lead us to consider an adaptive version of aSTDP in order to create a rule that is both competitive as well as rate stabilizing under different circumstances.

Next, we use a Fokker-Planck formalism to clarify what determines when an additive STDP rule fails to stabilize the postsynaptic firing rate. Here we derive the requirement for how the potentiation to depression ratio should change with neuronal activity. In the last section we provide a biologically realistic implementation of the adaptive rule and perform numerical simulations to show the how different parameterizations of the adaptive rule can guide STDP into differentially rate-sensitive regimes.

## 2 Additive STDP does not always stabilize learning

First, we numerically simulated an integrate-and-fire model receiving 1000 excitatory and 250 inhibitory afferents. The weights of the excitatory synapses were updated according to the additive STDP rule. We used the model developed by Song *et al*, 2000 [1]. The learning kernel $L(\tau)$ is $A_+ exp(\tau/\tau_+)$ if $\tau < 0$ or $-A_- \exp(-\tau/\tau_-)$ if $\tau > 0$ where $A_-/A_+$ denotes the amplitude of depression/potentiation respectively. Following [1] we use $\tau_+ = \tau_- = 20\ ms$ for the time window of learning. The integral over the temporal window of the synaptic learning function ($L$) is always negative. Synaptic weights change according to

$$\frac{dw_i}{dt} = \int L(\tau)s_{pre}(t+\tau)s_{post}(\tau)d\tau \ , \quad w_i\ \epsilon[0,\omega_{max}] \tag{1}$$

where $s(t)$ denotes a delta function representing a spike at time $t$. Correlations between input rates were generated by adding a common bias rate in a graded manner across synapses so that the first afferent is has zero while the last afferent has the maximal correlation, $C_{max}$.

We first examine how the depression/potentiation ratio ($\alpha = LTD/LTP$) [2] controls the dependence of the output firing rate on the synaptic input rate, here referred to as the effective neuronal gain. Provided that $\alpha$ is sufficiently large, the STDP rule controls the postsynaptic firing rate (Fig. 1A). The stabilizing effect of the STDP rule is therefore equivalent to having weak a neuronal gain.

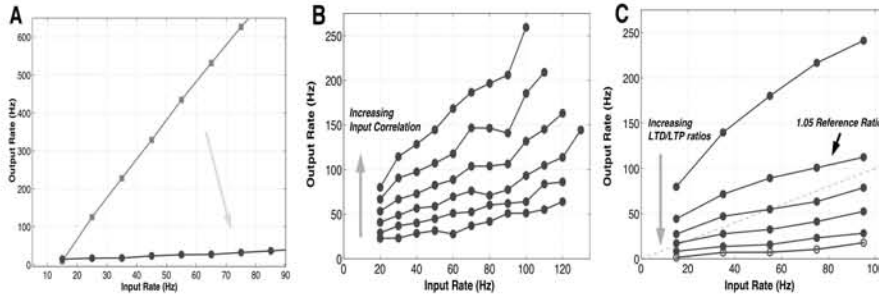

Figure 1: **A** STDP controls neuronal gain. The slope of the dependence of the postsynaptic output rate on the presynaptic input rate is referred to as the effective neuronal gain. The initial firing rate is shown by the upper curve while the lower line displays the final postsynaptic firing rate. The gain is reduced provided that the depression/potentiation ratio ($\alpha = 1.05$ here) is large enough. The input is uncorrelated. **B** Increasing input correlations increases neuronal gain. When the synaptic input is strongly correlated the postsynaptic neuron operates in a high gain mode characterized by a larger slope and larger baseline rate. Input correlations were uniformly distributed between 0 and a maximal value, $C_{max}$. The maximal correlation increases in the direction of the arrow: 0.0; 0.2; 0.3; 0.4; 0.5; 0.6; 0.7. The $\alpha$ ratio is 1.05. Note that for further increases in the presynaptic rates, postsynaptic firing can increase to over 1000 Hz. **C** The depression/potentiation ratio sets the neuronal gain. The $\alpha$ ratios increase in the direction of arrow:1.025;1.05;1.075;1.1025;1.155;1.2075. $C_{max}$ is 0.5.

We find that the neuronal gain is extremely sensitive to the value of $\alpha$ as well as to the amount of afferent input correlations. Figure 1B shows that increasing the amount of input correlations for a given $\alpha$ value increases the overall firing rate and the slope of the input-output curve, thus leading to larger effective gain. Increasing the amount of correlations between the synaptic afferents could therefore be interpreted as increasing the effective neuronal gain. Note that the baseline firing at a presynaptic drive of 20Hz is also increased. Next, we examined how neuronal gain depends on the value of $\alpha$ in the STDP rule (Figure 1C). The high gain and high rate mode induced by strong input correlations was reduced to a lower gain and lower rate mode by increasing $\alpha$ (see arrow in Figure 1C). Note, however, that there is no *correct* $\alpha$ value as it depends on both the input statistics as well as the desired input/output relationship.

# 3    Conditions for an adaptive additive STDP rule

Here we address how the learning ratio, $\alpha$, should depend on the input rate in order to produce a given neuronal input-output relationship. Using this functional form we will be able to formulate constraints for an adaptive additive STDP rule. This will guide us in the derivation of a biophysical implementation of the adaptive control scheme. The problem in its generality is to find (i) how the learning ratio should depend on the postsynaptic rate and (ii) how the postsynaptic rate depends on the input rate and the synaptic weights. By performing self-consistent calculations using a Fokker-Planck formulation, the problem is reduced to finding conditions for how the learning ratio should depend on the input rates only.

Let $\alpha$ denote depression/potentiation ratio $\alpha = LTD/LTP$ as before. Now we

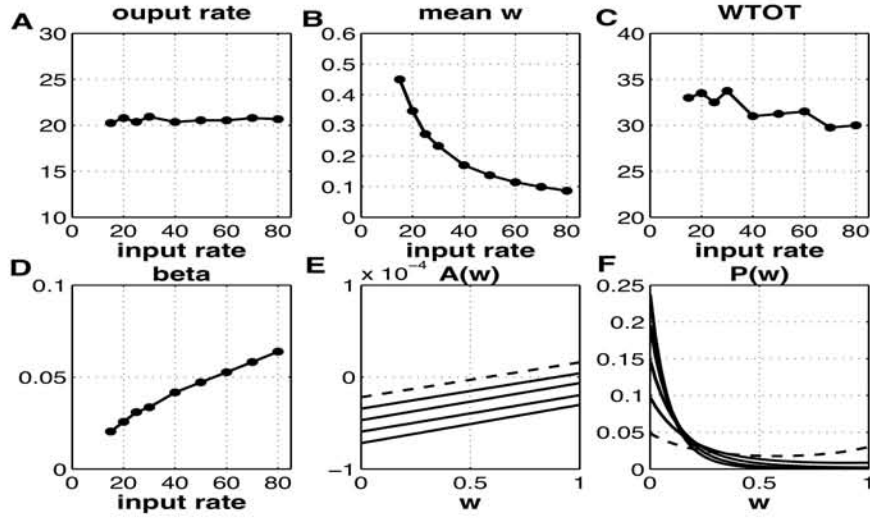

Figure 2: Self consistent Fokker-Planck calculations. Conditions for zero neuronal gain. U **A** The output rate does not depend on the input rate. Zero neuronal gain. **B** Dependence of the mean synaptic weight on input rates. **C** $W_{tot} \propto r_{pre} < w >$, see text. **D** The dependence of $\beta = \alpha - 1$ on input rate. **E,F** $A(w)$ and $P(w)$ are functions of the synaptic strength and depend on the input rate.. Note that eight different input rates are used but only traces 1, 3, 5, 7 are shown for $A(w)$ and $P(w)$ in which the dashed line correspond to the case with the lowest presynaptic rate.

determine how the parameter $\beta = \alpha - 1$ should scale with presynaptic rates in order to control the neuronal gain. The Fokker-Planck formulation permits an analytic calculation of the steady state distribution of synaptic weights [3]. The competition parameter for $N$ excitatory afferents is given by $W_{tot} = t_w r_{pre} N < w >$ where the time window $t_w$ is defined as the probability for depression ($p_d = t_w/t_{isi}$) that a synaptic event occurs within the time window ($t_w < t_{isi}$). The amount of potentiation and depression for the additive STDP yields in the steady-state, neglecting the exponential timing dependence, the following expression for the drift term $A(w)$

$$A(w) = p_d A_- [w/W_{tot} - (1 - 1/\alpha)] \qquad (2)$$

$A(w)$ represents the net weight "force field" experienced by an individual synapse. Thus, $A(w)$ determines whether a given synapse ($w$) will increase or decrease as a function of its synaptic weight. The steepness of the $A(w)$ function determines the degree of synaptic competition. The $w/W_{tot}$ is a competition term whereas the $(1 - 1/\alpha)$ provides a destabilizing force. When $w_{max} > (1 - 1/\alpha)W_{tot}$ the synaptic weight distribution is bimodal. The steady state distribution reads

$$P(w) = K e^{[(-w(1-1/\alpha)+w^2/(2W_{tot}))/(A_-)]} \qquad (3)$$

where K normalizes the $P(w)$ distribution [3].

Now, equations (2-3), with appropriate definitions of the terms, constitute a self-consistent system. Using these equations one can calculate how the parameter $\beta$

should scale with the presynaptic input rate in order to produce a given postsynaptic firing rate. For a given presynaptic rate, equations (2-3) can be iterated in until a self-consistent solution is found. At that point, the postsynaptic firing rate can be calculated. Here, instead we impose a fixed postsynaptic output rate for a given input rate and search for a self-consistent solution using $\beta$ as a free parameter. Performing this calculation for a range of input rates provides us with the desired dependency of $\beta$ on the presynaptic firing rate. Once a solution is reached we also examine the resulting steady state synaptic weight distribution ($P(w)$) and the corresponding drift term $A(w)$ as a function of the presynaptic input rate.

The results of such a calculation are illustrated in Figure 2. The neuronal gain, the ratio between the postsynaptic firing rate and the input rate is set to be zero (Fig 2A). To normalize postsynaptic firing rates the average synaptic weight has to decrease in order to compensate for the increasing presynaptic firing rate. This can be seen in (Fig 2B). The condition for a zero neuronal gain is that the average synaptic weight should decrease as $1/r_{pre}$. This makes $W_{tot}$ constant as shown in Fig 2C. For these values, $\beta$ has to increase with input rate as shown in Fig 2D. Note that this curve is approximately linear. The dependence of $A(w)$ and the synaptic weight distribution $P(w)$ on different presynaptic rates is illustrated in Fig 2E and F. As the presynaptic rates increase, the $A(w)$ function is lowered (dashed line indicates the smallest presynaptic rate), thus pushing more synapses to smaller values since they experience a net negative "force field". This is also reflected in the synaptic weight distribution which is pushed to the lower boundary as the input rates increase. When enforcing a different neuronal gain, the dependence of the $\beta$ term on the presynaptic rates remains approximately linear but with a different slope (not shown).

## 4    Derivation of an adaptive learning rule with biophysical components

The key insight from the above calculations is the observed linear dependence of $\beta$ on presynaptic rates. However, when implementing an adaptive rule with biophysical elements it is very likely that individual components will have a non-linear dependence on each other. The Fokker-Planck analysis suggests that the non-linearities should effectively cancel. Why should the system be linear? Another way to see from where the linearity requirement comes is that the $(w/W_{tot} - \beta)$ term in expression for $A(w)$ (valid for small $\beta$) has to be appropriately balanced when the input rates increases. The linearity of $\beta(r_{pre})$ follows from $W_{tot}$ being linear in $r_{pre}$.

Now, how could $\beta$ depend on presynaptic rates? A natural solution would be to use postsynaptic calcium to measure the postsynaptic firing and therefore indirectly the presynaptic firing rate. Moreover, the asymmetry ($\beta$) of the learning ratio could depend on the level of postsynaptic calcium. It is known that increased resting calcium levels inhibit NMDA channels and thus calcium influx due to synaptic input. Additionally, the calcium levels required for depression are easier to reach. Both of these effects in turn increase the probability of LTD induction. Incorporating these intermediate steps gives the following scheme:

$$\beta \overset{q}{\longleftrightarrow} Ca \overset{p}{\longleftrightarrow} r_{post} \overset{f_1}{\longleftrightarrow} r_{pre}$$

This scheme introduces parameters ($p$ and $q$) and a function ($f_1$) to control for the linearity/non-linearity between the variables. The global constraint from the Fokker-Planck is that the effective relation between $\beta$ and $r_{pre}$ should be linear. A biophysical formulation of the above scheme is the following

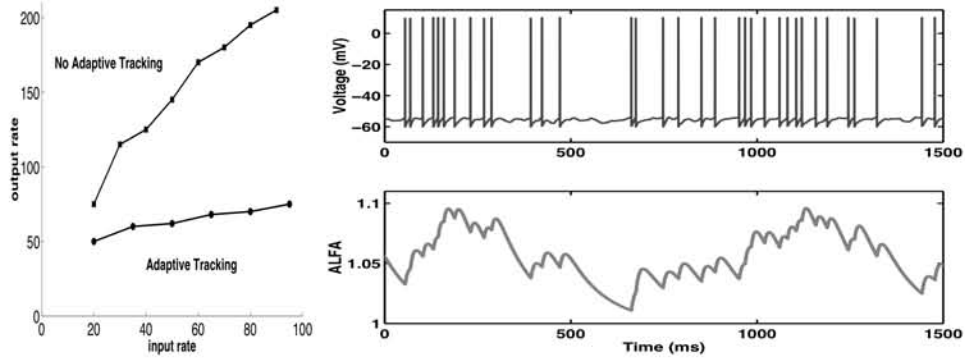

Figure 3: **Left** Steady-state response with (squares) or without (circles) the adaptive tracking scheme. When the STDP rule is extended with an adaptive control loop, the output rates are normalized in the presence of correlated input. **Right** Fast adaptive tracking. Since $\beta$ tracks changes in intracellular calcium on a rapid time-scale, every spike experiences a different learning ratio, $\alpha$. Note that the adaptive scheme approximates the learning ratio ($\alpha = 1.05$) used in [1].

$$\frac{d[Ca]}{dt} = -[Ca]/\tau_{Ca} + \gamma[\sum_i \delta(t - t_i)]^p \qquad (4)$$

$$\tau_\beta \frac{d\beta}{dt} = -\beta + [Ca]^q \qquad (5)$$

The parameter $p$ determines how the calcium concentration scales with the post-synaptic firing rate (delta spikes $\delta$ above) and $q$ controls the learning sensitivity. $\gamma$ controls the rise of steady-state calcium with increasing postsynaptic rates ($r_{post}$). The time constants $\tau_{Ca}$ and $\tau_\beta$ determine the calcium dynamics and the time course of the adaptive rule respectively. Note that we have not specified the neuronal transfer function, $f_1$.

To ensure a linear relation between $\beta$ and $r_{pre}$ it follows from the Fokker-Planck analysis that $[f_1(r_{pre})]^{pq}$ is approximately linear in $r_{pre}$. The neuronal gain can now be independently be controlled by the parameter $\gamma$. Moreover, the drift term $A(w)$ becomes

$$A(w) = p_d A_+ [w/W_{tot} - [\tau_{Ca}\gamma r^p_{post}]^q] \qquad (6)$$

for $\beta << 1$. $A(w)$ can be written in this form since we use that $w_d = -A_- = -A_+\alpha = -A_+(1 + [\tau_{Ca}\gamma r^p_{post}]^q)$. The $w/W_{tot}$ is a competition term whereas the $[\tau_{Ca}\gamma r^p_{post}]^q$ provides a destabilizing force. Note also, that when $w_{max} > [\tau_{Ca}\gamma r^p_{post}]^q W_{tot}$ there is a bimodal synaptic weight distribution and synaptic competition is preserved. A complete stability analysis is beyond the scope of the present study.

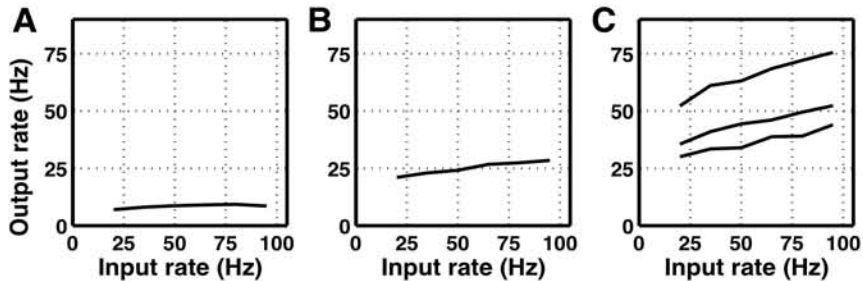

Figure 4: Full numerical simulation of the adaptive additive STDP rule. Parameters: $p = q = 1$. $\tau_{ca} = 10ms$, $\tau_\beta = 100ms$. **A** $\gamma = 1.25$. **B** $\gamma = 0.25$. **C** Input correlations are $C_{max} = 0, 0.3, 0.6$

## 5  Numerical simulations

Next, we examine whether the theory of adaptive normalization carry over to a full scale simulation of the integrate-and-fire model with the STDP rule and the biophysical adaptive scheme as described above. First, we studied the neuronal gain (cf. Figure 1) when the inputs were strongly correlated. Driving a neuron with increasing input rates increases the output rate significantly when there is no adaptive scheme (squares, Figure 3 Left) as observed previously (cf. Figure 1B). Adding the adaptive loop normalizes the output rates (circles, Figure 3 Left). This simulation shows that the average postsynaptic firing rate is regulated by the adaptive tracking scheme. This is expected since the Fokker-Planck analysis is based on the steady-state synaptic weight distribution. To further gain insight into the operation of the adaptive loop we examined the spike-to-spike dependence of the tracking scheme. Figure 3 (Right) displays the evolution of the membrane potential (top) and the learning ratio $\alpha = 1 + \beta$ (bottom). The adaptive rule tracks fast changes in firing by adjusting the learning ratio for each spike. Thus, the strength plasticity is different for every spike. Interestingly, the learning ratio ($\alpha$) fluctuates around the value 1.05 which was used in previous studies [1]. Our fast, spike-to-spike tracking scheme is in contrast to other homeostatic mechanisms operating on the time-scale of hours to days [11, 12, 13, 14]. In our formulation, the learning ratio, via $\beta$, tracks changes in intra-cellular calcium, which in turn reflects the instantaneous firing rate. Slower homeostatic mechanisms are unable to detect these rapid changes in firing statistics. Because this fast adaptive scheme depends on recent neuronal firing, pairing several spikes on the time-scale comparable to the calcium dynamics introduces non-linear summation effects.

Neurons with this adaptive STDP control loop can detect changes in the input correlation while being only weakly dependent on the presynaptic firing rate. Figure 4a and 4b show two different regimes corresponding to two different values of the parameter $\gamma$. In the high $\gamma$ regime (Fig. 4a) the neuronal gain is zero. The neuronal gain increased when $\gamma$ decreased (Fig. 4b) as expected from the theory. In a different regime where we introduce increasing correlations between the synaptic inputs [1] we find that the neuronal gain is changed little with increasing input rates but increases substantially with increasing input correlations (Fig 4c). Thus, the adaptive aSTDP rule can normalize the mean postsynaptic rate even when the input statistics change. With other adaptive parameters we also found learning regimes where the responses to input correlations were affected differentially (not shown).

# 6 Discussion

Synaptic learning rules have to operate under widely changes conditions such as different input statistics or neuromodulation. How can a learning rule dynamically guide a network into functionally similar operating regime under different conditions? We have addressed this issue in the context of spike-timing-dependent plasticity (STDP) [1, 10]. We found that STDP is very sensitive to the ratio of synaptic strengthening to weakening, $\alpha$, and requires different values for different input statistics. To correct for this, we proposed an adaptive control scheme to adjust the plasticity rule. This adaptive mechanisms makes the learning rule more robust to changing input conditions while preserving its interesting properties, such as synaptic competition. We suggested a biophysically plausible mechanism that can implement the adaptive changes consistent with the requirements derived using the Fokker-Planck analysis.

Our adaptive STDP rule adjusts the learning ratio on a millisecond time-scale. This in contrast to other, slow homeostatic controllers considered previously [11, 12, 13, 14, 3]. Because the learning rule changes rapidly, it is very sensitive the input statistics. Furthermore, the synaptic weight changes add non-linearly due to the rapid self-regulation. In recent experiments similar non-linearities have been detected (Y. Dan, personal communication) which might have roles in making synaptic plasticity adaptive. Finally, the new set of adaptive parameters could be independently controlled by meta-plasticity to bring the neuron into different operating regimes.

**Acknowledgments**

We thank Larry Abbott, Mark van Rossum, and Sen Song for helpful discussions. J.T. was supported by the Wennergren Foundation, and grants from Swedish Medical Research Foundation, and The Royal Academy for Science. A.K. was supported by the NIH Grant 2 R01 NS27337-12 and 5 R01 NS27337-13. Both A.K. and J.T. thank the Sloan Foundation for support.

# References

[1] Song, S., Miller, K., & Abbott, L. *Nature Neuroscience*, 3:919–926, 2000.

[2] Rubin, J., Lee, D., & Sompolinsky, H. *Physical Review Letter*, 86:364–367, 2001.

[3] van Rossum, M., G-Q, B., & Turrigiano, G. *J Neurosci*, 20:8812–8821, 2000.

[4] Sejnowski, T. *J Theoretical Biology*, 69:385–389, 1997.

[5] Abbott, L. & Nelson, S. *Nature Neuroscience*, 3:1178–1183, 2000.

[6] Miller, K. & MacKay, D. *Neural Computation*, 6:100–126, 1994.

[7] Markram, H., Lubke, J., Frotscher, M., & Sakmann, B. *Science*, 275:213–215, 1997.

[8] Bell, C., Han, V., Sugawara, Y., & Grant, K. *Nature*, 387:278–81, 1997.

[9] Bi, G.-Q. & Poo, M. *J Neuroscience*, 18:10464–10472, 1998.

[10] Kempter, R., Gerstner, W., & van Hemmen, J. *Neural Computation*, 13:2709–2742, 2001.

[11] Bell, A. In Moody, J., Hanson, S., & Lippmann, R., editors, *Advances in Neural Information Processing Systems*, volume 4. Morgan-Kaufmann, 1992.

[12] LeMasson, G., Marder, E., & Abbott, L. *Science*, 259:1915–7, 1993.

[13] Turrigiano, G., Leslie, K., Desai, N., Rutherford, L., & Nelson, S. *Nature*, 391:892–6, 1998.

[14] Turrigiano, G. & Nelson, S. *Curr Opin Neurobiol*, 10:358–64, 2000.
